# Shape Recipes: Scene Representations that Refer to the Image

**William T. Freeman and Antonio Torralba**
Artificial Intelligence Laboratory
Massachusetts Institute of Technology
Cambridge, MA 02139
`{wtf, torralba}@ai.mit.edu`

## Abstract

The goal of low-level vision is to estimate an underlying *scene*, given an observed image. Real-world scenes (eg, albedos or shapes) can be very complex, conventionally requiring high dimensional representations which are hard to estimate and store. We propose a low-dimensional representation, called a *scene recipe*, that relies on the image itself to describe the complex scene configurations. *Shape recipes* are an example: these are the regression coefficients that predict the bandpassed shape from image data. We describe the benefits of this representation, and show two uses illustrating their properties: (1) we improve stereo shape estimates by learning shape recipes at low resolution and applying them at full resolution; (2) Shape recipes implicitly contain information about lighting and materials and we use them for material segmentation.

## 1 Introduction

From images, we want to estimate various low-level scene properties such as shape, material, albedo or motion. For such an estimation task, the representation of the quantities to be estimated can be critical. Typically, these scene properties might be represented as a bitmap (eg [14]) or as a series expansion in a basis set of surface deformations (eg [10]). To represent accurately the details of real-world shapes and textures requires either full-resolution images or very high order series expansions. Estimating such high dimensional quantities is intrinsically difficult [2]. Strong priors [14] are often needed, which can give unrealistic shape reconstructions.

Here we propose a new scene representation with appealing qualities for estimation. The approach we propose is to let the image itself bear as much of the representational burden as possible. We assume that the image is always available and we describe the underlying scene *in reference to the image*. The scene representation is a set of rules for transforming from the local image information to the desired scene quantities. We call this representation a *scene recipe*: a simple function for transforming local image data to local scene data. The computer doesn't have to represent every curve of an intricate shape; the image does that for us, the computer just stores the rules for transforming from image to scene. In this paper, we focus on reconstructing the shapes that created the observed image, deriving *shape recipes*. The particular recipes we study here are regression coefficients for transforming

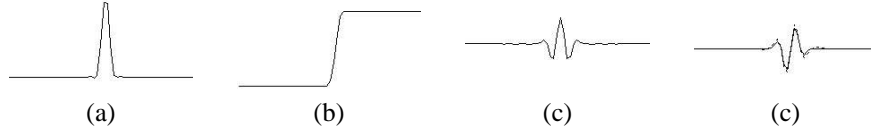

<div align="center">(a)           (b)           (c)           (c)</div>

Figure 1: 1-d example: The image (a) is rendered from the shape (b). The shape depends on the image in a non-local way. Bandpass filtering both signals allows for a local shape recipe. The dotted line (which agrees closely with true solid line) in (d) shows shape reconstruction from 9-parameter linear regression (9-tap convolution) from bandpassed image, (c).

bandpassed image data into bandpassed shape data.

## 2   Shape Recipes

The shape representation consists in describing, for a particular image, the functional relationship between image and shape. This relationship is not general for all images, but specific to the particular lighting and material conditions at hand. We call this functional relationship the *shape recipe*.

To simplify the computation to obtain shape from image data, we require that the scene recipes be local: the scene structure in a region should only depend on a local neighborhood of the image. It is easy to show that, without taking special care, the shape-image relationship is not local. Fig. 1 (a) shows the intensity profile of a 1-d image arising from the shape profile shown in Fig. 1 (b) under particular rendering conditions (a Phong model with 10% specularity). Note that the function to recover the shape from the image cannot be local because the identical local images on the left and right sides of the surface edge correspond to different shape heights.

In order to obtain locality in the shape-image relationship, we need to preprocess the shape and image signals. When shape and image are represented in a bandpass pyramid, within a subband, under generic rendering conditions [4], local shape changes lead to local image changes. (Representing the image in a Gaussian pyramid also gives a local relationship between image and bandpassed shape, effectively subsuming the image bandpass operation into the shape recipe. That formulation, explored in [16], can give slightly better performance and allows for simple non-linear extensions.) Figures 1 (c) and (d) are bandpass filtered versions of (a) and (b), using a second-derivative of a Gaussian filter. In this example, (d) relates to (c) by a simple shape recipe: convolution with a 9-tap filter, learned by linear regression from rendered random shape data. The solid line shows the true bandpassed shape, while the dotted line is the linear regression estimate from Fig. 1 (c).

For 2-d images, we break the image and shape into subbands using a steerable pyramid [13], an oriented multi-scale decomposition with non-aliased subbands (Fig. 3 (a) and (b)). A shape subband can be related to an image intensity subband by a function

$$Z_k = f_k(I_k) \tag{1}$$

where $f_k$ is a local function and $Z_k$ and $I_k$ are the $k$th subbands of the steerable pyramid representation of the shape and image, respectively. The simplest functional relationship between shape and image intensity is via a linear filter with a finite size impulse response: $Z_k \approx r_k \star I_k$, where $\star$ is convolution. The convolution kernel $r_k$ (specific to each scale and orientation) transforms the image subband $I_k$ into the shape subband $Z_k$. The recipe $r_k$ at each subband is learned by minimizing $\sum_x |Z_k - I_k \star r_k|^2$, regularizing $r_k$ as needed to avoid overfitting. $r_k$ contains information about the particular lighting conditions and the surface material. More general functions can be built by using non-linear filters and combining image information from different orientations and scales [16].

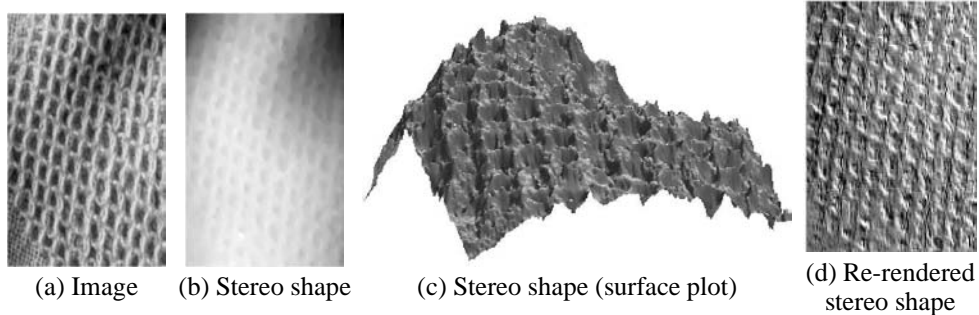

(a) Image    (b) Stereo shape    (c) Stereo shape (surface plot)    (d) Re-rendered stereo shape

Figure 2: Shape estimate from stereo. (a) is one image of the stereo pair; the stereo reconstruction is depicted as (b) a range map and (c) a surface plot and (d) a re-rendering of the stereo shape. The stereo shape is noisy and misses fine details.

We conjecture that multiscale shape recipes have various desirable properties for estimation. First, they allow for a compact encoding of shape information, as much of the complexity of the shape is encoded in the image itself. The recipes need only specify how to translate image into shape. Secondly, regularities in how the shape recipes $f_k$ vary across scale and space provide a powerful mechanism for regularizing shape estimates. Instead of regularizing shape estimates by assuming a prior of smoothness of the surface, we can assume a slow spatial variation of the functional relationship between image and shape, which should make estimating shape recipes easier. Third, shape recipes implicitly encode lighting and material information, which can be used for material-based segmentation. In the next two sections we discuss the properties of smoothness across scale and space and we show potential applications in improving shape estimates from stereo and in image segmentation based on material properties.

## 3   Scaling regularities of shape recipes

Fig. 2 shows one image of a stereo pair and the associated shape estimated from a stereo algorithm[1]. The shape estimate is noisy in the high frequencies (see surface plot and re-rendered shape), but we assume it is accurate in the low spatial frequencies.

Fig. 3 shows the steerable pyramid representations of the image (a) and shape (b) and the learned shape recipes (c) for each subband (linear convolution kernels that give the shape subband from the image subband). We exploit the slow variation of shape recipes over scale and assume that the shape recipes are *constant* over the top four octaves of the pyramid[2] Thus, from the shape recipes learned at low-resolution we can reconstruct a higher resolution shape estimate than the stereo output, by learning the rendering conditions then taking advantage of shape details visible in the image but not exploited by the stereo algorithm. Fig. 4 (a) and (b) show the image and the implicit shape representation: the pyramid's low-resolution shape and the shape recipes used over the top four scales. Fig. 4 (c) and (d) show explicitly the reconstructed shape implied by (a) and (b): note the high resolution details, including the fine structure visible in the bottom left corner of (d). Compare with the stereo

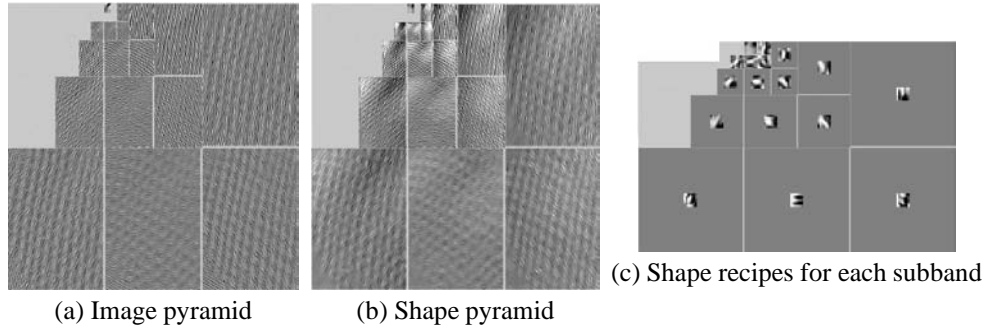

(a) Image pyramid (b) Shape pyramid (c) Shape recipes for each subband

Figure 3: Learning shape recipes at each subband. (a) and (b) are the steerable pyramid representations [13] of image and stereo shape. (c) shows the convolution kernels that best predict (b) from (a). The steerable pyramid isolates information according to scale (the smaller subband images represent larger spatial scales) and orientation (clockwise among subbands of one size: vertical, diagonal, horizontal, other diagonal).

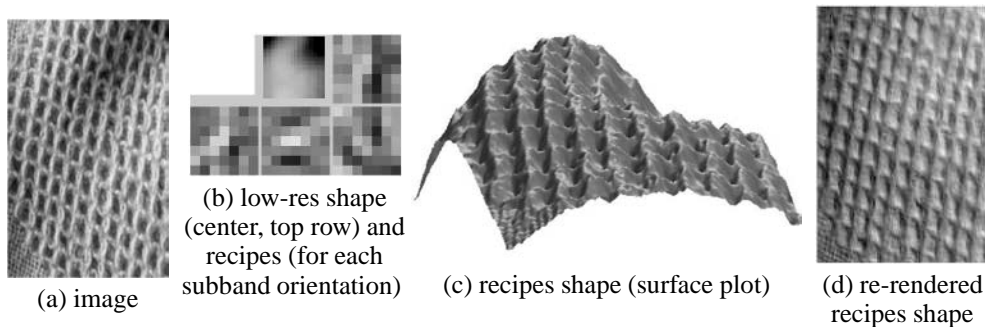

(a) image
(b) low-res shape (center, top row) and recipes (for each subband orientation)
(c) recipes shape (surface plot)
(d) re-rendered recipes shape

Figure 4: Reconstruction from shape recipes. The shape is represented by the information contained in the image (a), the low-res shape pyramid residual and the shape recipes (b) estimated at the lowest resolution. The shape can be regenerated by applying the shape recipes (b) at the 4 highest resolution scales, then reconstructing from the shape pyramid. (d) shows the image re-rendered under different lighting conditions than (a). The reconstruction is not noisy and shows more detail than the stereo shape, Fig. 2, including the fine textures visible at the bottom left of the image (a) but not detected by the stereo algorithm.

output in Fig. 2.

## 4 Segmenting shape recipes

Segmenting an image into regions of uniform color or texture is often an approximation to an underlying goal of segmenting the image into regions of uniform material. Shape recipes, by describing how to transform from image to shape, implicitly encode both lighting and material properties. Across unchanging lighting conditions, segmenting by shape recipes allows us to segment according to a material's rendering properties, even overcoming changes of intensities or texture of the rendered image. (See [6] for a non-parametric approach to material segmentation.)

We expect shape recipes to vary smoothly over space except for abrupt boundaries at changes in material or illumination. Within each subband, we can write the shape $Z_k$

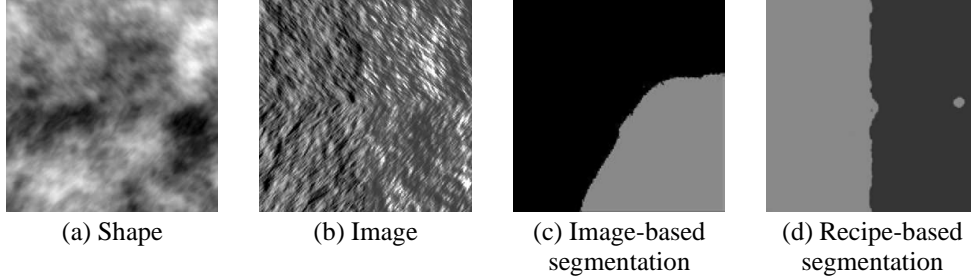

| (a) Shape | (b) Image | (c) Image-based segmentation | (d) Recipe-based segmentation |

Figure 5: Segmentation example. Shape (a), with a horizontal orientation discontinuity, is rendered with two different shading models split vertically, (b). Based on image information alone, it is difficult to find a good segmentation into 2 groups, (c). A segmentation into 2 different shape recipes naturally falls along the vertical material boundary, (d).

as a mixture of recipes:

$$p(Z_k|I_k) = \sum_{n=1}^{N} p(Z_k - f_{k,n}(I_k))p_n \qquad (2)$$

where $N$ specifies the number of recipes needed to explain the underlying shape $Z_k$. The weights $p_n$, which will be a function of location, will specify which recipe has to be used within each region and, therefore, will provide a segmentation of the image.

To estimate the parameters of the mixture (shape recipes and weights), given known shape and the associated image, we use the EM algorithm [17]. We encourage spatial continuity for the weights $p_n$ as neighboring pixels are likely to belong to the same material. We use the mean field approximation to implement the spatial smoothness prior in the E step, suggested in [17].

Figure 5 shows a segmentation example. (a) is a fractal shape, with diagonal left structure across the top half, and diagonal right structure across the bottom half. Onto that shape, we "painted" two different Phong shading renderings in the two vertical halves, shown in (b) (the right half is shinier than the left). Thus, texture changes in each of the four quadrants, but the only material transition is across the vertical centerline. An image-based segmentation, which makes use of texture and intensity cues, among others, finds the four quadrants when looking for 4 groups, but can't segment well when forced to find 2 groups, (c). (We used the normalized cuts segmentation software, available on-line [11].) The shape recipes encode the *relationship* between image and shape when segmenting into 2 groups, and finds the vertical material boundary, (d).

## 5 Occlusion boundaries

Not all image variations have a direct translation into shape. This is true for paint boundaries and for most occlusion boundaries. These cases need to be treated specially with shape recipes. To illustrate, in Fig. 6 (c) the occluding boundary in the shape only produces a smooth change in the image, Fig. 6 (a). In that region, a shape recipe will produce an incorrect shape estimate, however, the stereo algorithm will often succeed at finding those occlusion edges. On the other hand, stereo often fails to provide the shape of image regions with complex shape details, where the shape recipes succeed. For the special case of revising the stereo algorithm's output using shape recipes, we propose a statistical framework to combine both sources of information. We want to estimate the shape $Z$ that maximizes the likelihood given the shape from stereo $S$ and shape from image intensity $I$

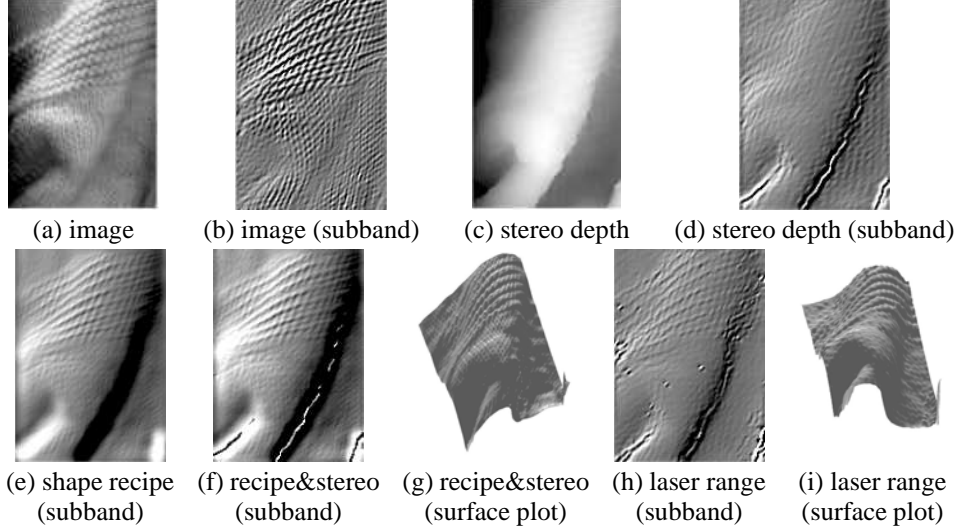

(a) image     (b) image (subband)     (c) stereo depth     (d) stereo depth (subband)

(e) shape recipe (subband)    (f) recipe&stereo (subband)    (g) recipe&stereo (surface plot)    (h) laser range (subband)    (i) laser range (surface plot)

Figure 6: One way to handle occlusions with shape recipes. Image in full-res (a) and one steerable pyramid subband (b); stereo depth, full-res (c) and subband (d). (e) shows subband of shape reconstruction using learned shape recipe. Direct application of shape recipe across occlusion boundary misses the shape discontinuity. Stereo algorithm catches that discontinuity, but misses other shape details. Probabilistic combination of the two shape estimates (f, subband, g, surface), assuming Laplacian shape statistics, captures the desirable details of both, comparing favorably with laser scanner ground truth, (h, subband, i, surface, at slight misalignment from photos).

via shape recipes:

$$p(Z|S, I) = p(S, I|Z)p(Z)/p(S, I) \tag{3}$$

(For notational simplicity, we omit the spatial dependency from $I$, $S$ and $Z$.) As both stereo $S$ and image intensity $I$ provide strong constraints for the possible underlying shape $Z$, the factor $p(Z)$ can be considered constant in the region of support of $p(S, I|Z)$. $p(S, I)$ is a normalization factor. Eq. (3) can be simplified by assuming that the shapes from stereo and from shape recipes are independent. Furthermore, we also assume independence between the pixels in the image and across subbands:

$$p(S, I|Z) = \prod_k \prod_{x,y} p(S_k|Z_k)p(I_k|Z_k) \tag{4}$$

$S_k$, $Z_k$ and $I_k$ refer to the outputs of the subband $k$. Although this is an oversimplification it simplifies the analysis and provides good results.

The terms $p(S_k|Z_k)$ and $p(I_k|Z_k)$ will depend on the noise models for the depth from stereo and for the shape recipes. For the shape estimate from stereo we assume a Gaussian distribution for the noise. At each subband and spatial location we have:

$$p(S_k|Z_k) = p_s(Z_k - S_k) = \frac{e^{-|Z_k - S_k|^2/\sigma_s^2}}{(2\pi)^{1/2}\sigma_s} \tag{5}$$

In the case of the shape recipes, a Gaussian noise model is not adequate. The distribution of the error $Z_k - f_k(I_k)$ will depend on image noise, but more importantly, on all shape and image variations that are not functionally related with each other through the recipes. Fig. 6 illustrates this point: the image data, Fig. 6 (b) does not describe the discontinuity that

exists in the shape, Fig. 6(h). When trying to estimate shape using the shape recipe $f_k(I_k)$, it fails to capture the discontinuity although it captures correctly other texture variations, Fig. 6 (e). Therefore, $Z_k - f_k(I_k)$ will describe the distribution of occluding edges that do not produce image variations and paint edges that do not translate into shape variations. Due to the sparse distribution of edges in images (and range data), we expect $Z_k - f_k(I_k)$ to have a Laplacian distribution typical of the statistics of wavelet outputs of natural images [12]:

$$p(I_k|Z_k) = p(Z_k - f_k(I_k)) = \frac{e^{-|Z_k - f_k(I_k)|^p/\sigma_i^p}}{2\sigma_i/p\Gamma(1/p)} \qquad (6)$$

In order to verify this, we use the stereo information at the low spatial resolutions that we expect is correct so that: $p(Z_k - f_k(I_k)) \simeq p(S_k - f_k(I_k))$. We obtain values of $p$ in the range $(0.6, 1.2)$. We set $p = 1$ for the results shown here. Note that $p = 2$ gives a Gaussian distribution.

The least square estimate for the shape subband $Z_k$ given both stereo and image data, is:

$$\hat{Z}_k = \int Z_k p(Z_k|S_k, I_k)dZ_k = \frac{\int Z_k p(S_k|Z_k)p(I_k|Z_k)dZ_k}{\int p(S_k|Z_k)p(I_k|Z_k)dZ_k} \qquad (7)$$

This integral can be evaluated numerically independently at each pixel. When $p = 2$, then the LSE estimation is a weighted linear combination of the shape from stereo and shape recipes. However, with $p \simeq 1$ this problem is similar to the one of image denosing from wavelet decompositions [12] providing a non-linear combination of stereo and shape recipes. The basic behavior of Eq. (7) is to take from the stereo everything that cannot be explained by the recipes, and to take from the recipes the rest. Whenever both stereo and shape recipes give similar estimates, we prefer the recipes because they are more accurate than the stereo information. Where stereo and shape recipes differ greatly, such as at occlusions, then the shape estimate follows the stereo shape.

## 6  Discussion and Summary

Unlike shape-from-shading algorithms [5], shape recipes are *fast, local* procedures for computing shape from image. The approximation of linear shading [7] also assumes a local linear relationship between image and shape subbands. However, learning the regression coefficients allows a linearized fit to more general rendering conditions than the special case of Lambertian shading for which linear shading was derived.

We have proposed shape recipes as a representation that leaves the burden of describing shape details to the image. Unlike many other shape representations, these are low-dimensional, and should change slowly over time, distance, and spatial scale. We expect that these properties will prove useful for estimation algorithms using these representations, including non-linear extensions [16].

We showed that some of these properties are indeed useful in practice. We developed a shape estimate improver that relies on an initial estimate being accurate at low resolutions. Assuming that a shape recipes change slowly over 4 octaves of spatial scale, we learned the shape recipes at low resolution and applied them at high resolution to find shape from image details not exploited by the stereo algorithm. Comparisons with ground truth shapes show good results. Shape recipes fold in information about both lighting and material properties and can also be used to estimate material boundaries over regions where the lighting is assumed to be constant.

Gilchrist and Adelson describe "atmospheres", which are local formulas for converting image intensities to perceived lightness values [3, 1]. In this framework, atmospheres are "lightness recipes". A full description of an image in terms of a scene recipe would require both shape recipes and reflectance recipes (for computing reflectance values from

image data), which also requires labelling parts of the image as being caused by shading or reflectance changes, such as [15].

At a conceptual level, this representation is consistent with a theme in human vision research, that our visual systems use the world as a framebuffer or visual memory, not storing in the brain what can be obtained by looking [9]. Using shape recipes, we find simple transformation rules that let us convert from image to shape whenever we need to, by examining the image.

*We thank Ray Jones and Leonard McMillan for providing Cyberware scans, and Hao Zhang for code for rectification of stereo images. This work was funded by the Nippon Telegraph and Telephone Corporation as part of the NTT/MIT Collaboration Agreement.*

## Footnotes

[1]We took our stereo photographs using a 3.3 Megapixel Olympus Camedia C-3040 camera, with a Pentax stereo adapter. We calibrated the stereo images using the point matching algorithm of Zhang [18], and rectified the stereo pair (so that epipoles are along scan lines) using the algorithm of [8], estimating disparity with the Zitnick–Kanade stereo algorithm [19].

[2]Except for a scale factor. We scale the amplitude of the fixed recipe convolution kernels by 2 for each octave, to account for the differentiation operation in the linear shading approximation to Lambertian rendering [7].

## References

[1] E. H. Adelson. Lightness perception and lightness illusions. In M. Gazzaniga, editor, *The New Cognitive Neurosciences*, pages 339–351. MIT Press, 2000.

[2] C. M. Bishop. *Neural networks for pattern recognition*. Oxford, 1995.

[3] A. Gilchrist et al. An anchoring theory of lightness. *Psychological Review*, 106(4):795–834, 1999.

[4] W. T. Freeman. The generic viewpoint assumption in a framework for visual perception. *Nature*, 368(6471):542–545, April 7 1994.

[5] B. K. P. Horn and M. J. Brooks, editors. *Shape from shading*. The MIT Press, Cambridge, MA, 1989.

[6] T. Leung and J. Malik. Representing and recognizing the visual appearance of materials using three-dimensional textons. *Intl. J. Comp. Vis.*, 43(1):29–44, 2001.

[7] A. P. Pentland. Linear shape from shading. *Intl. J. Comp. Vis.*, 1(4):153–162, 1990.

[8] M. Pollefeys, R. Koch, and L. V. Gool. A simple and efficient rectification method for general motion. In *Intl. Conf. on Computer Vision (ICCV)*, pages 496–501, 1999.

[9] R. A. Rensink. The dynamic representation of scenes. *Vis. Cognition*, 7:17–42, 2000.

[10] S. Sclaroff and A. Pentland. Generalized implicit functions for computer graphics. In *Proc. SIGGRAPH 91*, volume 25, pages 247–250, 1991. In *Computer Graphics*, Annual Conference Series.

[11] J. Shi and J. Malik. Normalized cuts and image segmentation. *IEEE Pattern Analysis and Machine Intelligence*, 22(8):888–905, 2000.

[12] E. P. Simoncelli. Statistical models for images: Compression, restoration and synthesis. In *31st Asilomar Conf. on Sig., Sys. and Computers*, Pacific Grove, CA, 1997.

[13] E. P. Simoncelli and W. T. Freeman. The steerable pyramid: a flexible architecture for multi-scale derivative computation. In *2nd Annual Intl. Conf. on Image Processing*, Washington, DC, 1995. IEEE.

[14] R. Szeliski. Bayesian modeling of uncertainty in low-level vision. *Intl. J. Comp. Vis.*, 5(3):271–301, 1990.

[15] M. F. Tappen, W. T. Freeman, and E. H. Adelson. Recovering intrinsic images from a single image. In *Adv. in Neural Info. Proc. Systems*, volume 15. MIT Press, 2003.

[16] A. Torralba and W. T. Freeman. Properties and applications of shape recipes. Technical Report AIM-2002-019, MIT AI lab, 2002.

[17] Y. Weiss. *Bayesian motion estimation and segmentation*. PhD thesis, M.I.T., 1998.

[18] Z. Zhang. Determining the epipolar geometry and its uncertainty: A review. Technical Report 2927, Sophia-Antipolis Cedex, France, 1996. see http://www-sop.inria.fr/robotvis/demo/f-http/html/.

[19] C. L. Zitnick and T. Kanade. A cooperative algorithm for stereo matching and occlusion detection. *IEEE Pattern Analysis and Machine Intelligence*, 22(7), July 2000.
